# Theories Of Access Consciousness

**Michael D. Colagrosso**
Department of Computer Science
Colorado School of Mines
Golden, CO 80401 USA
mcolagro@mines.edu

**Michael C. Mozer**
Institute of Cognitive Science
University of Colorado
Boulder, CO 80309 USA
mozer@colorado.edu

## Abstract

Theories of access consciousness address how it is that some mental states but not others are available for evaluation, choice behavior, and verbal report. Farah, O'Reilly, and Vecera (1994) argue that quality of representation is critical; Dehaene, Sergent, and Changeux (2003) argue that the ability to communicate representations is critical. We present a *probabilistic information transmission* or *PIT* model that suggests both of these conditions are essential for access consciousness. Having successfully modeled data from the repetition priming literature in the past, we use the PIT model to account for data from two experiments on subliminal priming, showing that the model produces priming even in the absence of accessibility and reportability of internal states. The model provides a mechanistic basis for understanding the dissociation of priming and awareness.

Philosophy has made many attempts to identify distinct aspects of consciousness. Perhaps the most famous effort is Block's (1995) delineation of phenomenal and access consciousness. Phenomenal consciousness has to do with "what it is like" to experience chocolate or a pin prick. Access consciousness refers to internal states whose content is "(1) inferentially promiscuous, i.e., poised to be used as a premise in reasoning, (2) poised for control of action, and (3) poised for rational control of speech." (p. 230) The scientific study of consciousness has exploded in the past six years, and an important catalyst for this explosion has been the decision to focus on the problem of access consciousness: how is it that some mental states but not others become available for evaluation, choice behavior, verbal report, and storage in working memory. Another reason for the recent explosion of consciousness research is the availability of functional imaging techniques to explore differences in brain activation between conscious and unconscious states, as well as the development of clever psychological experiments that show that a stimulus that is not consciously perceived can nonetheless influence cognition, which we describe shortly.

## 1   Subliminal Priming

The phenomena we address utilize an experimental paradigm known as *repetition priming*. Priming refers to an improvement in efficiency in processing a stimulus item as a result of previous exposure to the item. Efficiency is defined in terms of shorter response times, lower error rates, or both. A typical long-term perceptual priming experiment consists of a study phase during which participants are asked to read aloud a list of words, and a test phase during which participants must name or categorize a series of words, presented one at a time. Reaction time is lower and/or accuracy is higher for test words that were also on the study list. Repetition priming occurs without strategic effort on the part of participants, and therefore appears to be a low level mechanism of learning, which likely serves as the mechanism underlying the refinement of cognitive skills with practice.

In traditional studies, priming is *supraliminal*—the prime is consciously perceived. In the studies we model here, primes are *subliminal*. Subliminal priming addresses fundamental issues concerning conscious access: How is it that a word or image that cannot be identified, detected, or even discriminated in forced choice can nonetheless influence the processing of a subsequent stimulus word? Answering this question in a computational framework would be a significant advance toward understanding the nature of access consciousness.

## 2 Models of Conscious and Unconscious Processing

In contrast to the wealth of experimental data, and the large number of speculative and philosophical papers on consciousness, concrete computational models are rare. The domain of consciousness is particularly ripe for theoretical perspectives, because it is a significant contribution to simply provide an existence proof of a mechanism that can explain specific experimental data. Ordinarily, a theorist faces skepticism when presenting a model; it often seems that hundreds of alternative, equally plausible accounts must exist. However, when addressing data deemed central to issues of consciousness, simply providing a concrete handle on the phenomena serves to demystify consciousness and bring it into the realm of scientific understanding.

We are familiar with only three computational models that address specific experimental data in the domain of consciousness. We summarize these models, and then present a novel model and describe its relationship to the previous efforts. Farah, O'Reilly, and Vecera (1994) were the first to model specific phenomena pertaining to consciousness in a computational framework. The phenomena involve prosopagnosia, a deficit of overt face recognition following brain damage. Nonetheless, prosopagnosia patients exhibit residual covert recognition by a variety of tests. For example, when patients are asked to categorize names as famous or nonfamous, their response times are faster to a famous name when the name is primed by a picture of a semantically related face (e.g., the name "Bill Clinton" when preceded by a photograph of Hillary), despite the fact that they could not identify the related face. Farah et al. model face recognition in a neural network, and show that when the network is damaged, it loses the ability to perform tasks requiring high fidelity representations (e.g., identification) but not tasks requiring only coarse information (e.g., semantic priming). They argue that conscious perception is associated with a certain minimal quality of representation.

Dehaene and Naccache (2001) outline a framework based on Baars' (1989) notion of conscious states as residing in a global workspace. They describe the workspace as a "distributed neural system...with long-distance connectivity that can potentially interconnect multiple specialized brain areas in a coordinated, though variable manner." (p. 13) Dehaene, Sergent, and Changeaux (2003) implement this framework in a complicated architecture of integrate-and-fire neurons and show that the model can qualitatively account for the *attentional blink* phenomenon. The attentional blink is observed in experiments where participants are shown a rapid series of stimuli, which includes two targets (T1 and T2). If T2 appears shortly after T1, the ability to report T2 drops, as if attention is distracted. Dehane et al. explain this phenomenon as follows. When T1 is presented, its activation propagates to frontal cortical areas (the global workspace). Feedback connections lead to a resonance between frontal and posterior areas, which strengthen T1 but block T2 from entering the workspace. If the T1-T2 lag is sufficiently great, habituation of T1 sufficiently weakens the representation such that T2 can enter the workspace and suppress T1. In this account, conscious access is achieved via resonance between posterior and frontal areas.

Although the Farah et al. and Dehaene et al. models might not seem to have much in common, they both make claims concerning what is required to achieve functional connectivity between perceptual and response systems. Farah et al. focus on aspects of the representation; Dehaene et al. focus on a pathway through which representations can be

communicated. These two aspects are not incompatible, and in fact, a third model incorporates both. Mathis and Mozer (1996) describe an architecture with processing modules for perceptual and response processes, implemented as attractor neural nets. They argue that in order for a representation in some perceptual module to be assured of influencing a response module, (a) it must have certain characteristics–temporal persistence and well-formedness– which is quite similar to Farah et al.'s notion of quality, and (b) the two modules must be interconnected—which is the purpose of Dehaene et al.'s global workspace. The model has two limitations that restrict its value as a contemporary account of conscious access. First, it addressed classical subliminal priming data, but more reliable data has recently been reported. Second, like the other two models, Mathis and Mozer used a complex neural network architecture with arbitrary assumptions built in, and the sensitivity of the model's behavior to these assumptions is far from clear. In this paper, we present a model that embodies the same assumptions as Mathis and Mozer, but overcomes its two limitations, and explains subliminal-priming data that has yet to be interpreted via a computational model.

## 3  The Probabilistic Information Transmission (PIT) Framework

Our model is based on the *probabilistic information transmission* or *PIT* framework of Mozer, Colagrosso, and Huber (2002, 2003). The framework characterizes the transmission of information from perceptual to response systems, and how the time course of information transmission changes with experience (i.e., priming). Mozer et al. used this framework to account for a variety of facilitation effects from supraliminal repetition priming.

The framework describes cognition in terms of a collection of information-processing *pathways*, and supposes that any act of cognition involves coordination among multiple pathways. For example, to model a letter-naming task where a letter printed in upper or lower case is presented visually and the letter must be named, the framework would assume a *perceptual pathway* that maps the visual input to an identity representation, and a *response pathway* that maps a identity representation to a naming response. The framework is formalized as a probabilistic model: the pathway input and output are random variables and microinference in a pathway is carried out by Bayesian belief revision.

The framework captures the time course of information processing for a single experimental trial. To elaborate, consider a pathway whose input at time $t$ is a discrete random variable, denoted $X(t)$, which can assume values $x_1, x_2, x_3, \ldots, x_{n_x}$ corresponding to alternative input states. Similarly, the output of the pathway at time $t$ is a discrete random variable, denoted $Y(t)$, which can assume values $y_1, y_2, y_3, \ldots, y_{n_y}$. For example, in the letter-naming task, the input to the perceptual pathway would be one of $n_x = 52$ visual patterns corresponding to the upper- and lower-case letters of the alphabet, and the output is one of $n_y = 26$ letter identities. To present a particular input alternative, say $x_i$, to the model for $T$ time steps, we specify $X(t) = x_i$ for $t = 1 \ldots T$, and allow the model to compute $P(Y(t) \mid X(1) \ldots X(t))$.

A pathway is modeled as a dynamic Bayes network; the minimal version of the model used in the present simulations is simply a hidden Markov model, where the $X(t)$ are observations and the $Y(t)$ are inferred state (see Figure 1a). In typical usage, an HMM is presented with a sequence of distinct inputs, whereas we maintain the same input for many successive time steps; and an HMM transitions through a sequence of distinct hidden states, whereas we attempt to converge with increasing confidence on a single state.

Figure 1b illustrates the time course of inference in a single pathway with 52 input and 26 output alternatives and two-to-one associations. The solid line in the Figure shows, as a function of time $t$, $P(Y(t) = y_i \mid X(1) = x_{2i} \ldots X(t) = x_{2i})$, i.e., the probability that input $i$ (say, the visual pattern of an upper case O) will produce its target output (the letter identity). Evidence for the target output accumulates gradually over time, yielding a speed-accuracy curve that relates the number of iterations to the accuracy of identification.

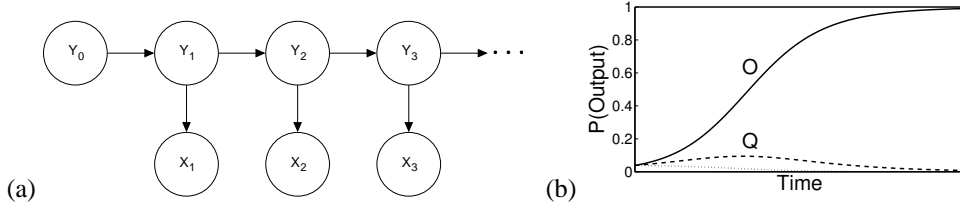

(a)                                                                                          (b)

Figure 1: (a) basic pathway architecture—a hidden Markov model; (b) time course of inference in a
pathway when the letter O is presented, causing activation of both O and the visually similar Q.

The exact shape of the speed-accuracy curve—the pathway dynamics—are determined by
three probability distributions, which embody the knowledge and past experience of the
model. First, $P(Y(0))$ is the *prior* distribution over outputs in the absence of any informa-
tion about the input. Second, $P(Y(t) \mid Y(t-1))$ characterizes how the pathway output
evolves over time. We assume the transition probability matrix serves as a memory with
diffusion, i.e., $P(Y(t) = y_i|Y(t-1) = y_j) = (1-\beta)\delta_{ij} + \beta P(Y(0) = y_i)$, where $\beta$ is the
diffusion constant and $\delta_{ij}$ is the Kronecker delta. Third, $P(X(t) \mid Y(t))$ characterizes the
*strength of association* between inputs and outputs. The greater the association strength,
the more rapidly that information about $X$ will be communicated to $Y$. We parameterize
this distribution as $P(X(t) = x_i|Y(t) = y_j) \sim 1 + \sum_k \gamma_{ik}\alpha_{kj}$, where $\alpha_{ij}$ indicates the
frequency of experience with the association between states $x_i$ and $y_j$, and $\gamma_{ik}$ specifies the
similarity between states $x_i$ and $x_k$. (Although the representation of states is localist, the $\gamma$
terms allow us to design in the similarity structure inherent in a distributed representation.)
These association strengths are highly constrained by the task structure and the similarity
structure and familiarity of the inputs.

Fundamental to the framework is the assumption that with each experience, a pathway be-
comes more efficient at processing an input. Efficiency is reflected by a shift in the speed-
accuracy curve to the left. In Mozer, Colagrosso, and Huber (2002, 2003), we propose two
distinct mechanisms to model phenomena of supraliminal priming. First, the association
frequencies, $\alpha_{ij}$, are increased following a trial in which $x_i$ leads to activation of $y_j$, re-
sulting in more efficient transmission of information, corresponding to an increased slope
of the solid line in Figure 1b. The increase is Hebbian, based on the maximum activation
achieved by $x_i$ and $y_j$: $\Delta\alpha_{ij} = \eta \max_t P(X(t) = x_i)P(Y(t) = y_j)$, where $\eta$ is a step size.
Second, the priors, which serve as a model of the environment, are increased to indicate a
greater likelihood of the same output occurring again in the future. In modeling data from
supraliminal priming, we found that the increases to association frequencies are long last-
ing, but the increases to the priors decay over the course of a few minutes or a few trials.
As a result, the prior updating does not play into the simulation we report here; we refer
the reader to Mozer, Colagrosso, and Huber (2003) for details.

## 4  Access Consciousness and PIT

We have described the operation of a single pathway, but to model any cognitive task,
we require a series of pathways in cascade. For a simple choice task, we use a percpet-
ual pathway cascaded to a response pathway. The interconnection between the pathways
is achieved by copying the output of the perceptual pathway, $Y^p(t)$, to the input of the
response pathway, $X^r(t)$, at each time $t$.

This multiple-pathway architecture allows us to characterize the notion of access con-
sciousness. Considering the output of the perceptual pathway, access is achieved when:
(1) the output representation is sufficient to trigger the correct behavior in the response
pathway, and (2) the perceptual and response pathways are functionally interconnected. In
more general terms, access for a perceptual pathway output requires that these two condi-

tions be met not just for a specific response pathway, but for arbitrary response pathways (e.g., pathways for naming, choice, evaluation, working memory, etc.). In Mozer and Colagrosso (in preparation) we characterize the sufficiency requirements of condition 1; they involve a representation of low entropy that stays active for long enough that the representation can propagate to the next pathway.

As we will show, a briefly presented stimulus fails to achieve a representation that supports choice and naming responses. Nonetheless, the stimulus evokes activity in the perceptual pathway. Because perceptual priming depends on the magnitude of the activation in the perceptual pathway, not on the activation being communicated to response pathways, the framework is consistent with the notion of priming occurring in the absence of awareness.

### 4.1 Simulation of Bar and Biederman (1998)

Bar and Biederman (1998) presented a sequence of masked line drawings of objects and asked participants to name the objects, even if they had to guess. If the guess was incorrect, participants were required to choose the object name from a set of four alternatives. Unbeknownst to the participant, some of the drawings in the series were repeated, and Bar and Biederman were interested in whether participants would benefit from the first presentation even if it could not be identified. The repeated objects could be the same or a different exemplar of the object, and it could appear in either the same or a different display position.

Participants were able to name 13.5% of drawings on presentation 1, but accuracy jumped to 34.5% on presentation 2. Accuracy did improve, though not as much, if the same shape was presented in a different position, but not if a different drawing of the same object was presented, suggesting a locus of priming early in the visual stream. The improvement in accuracy is not due to practice in general, because accuracy rose only 4.0% for novel control objects over the course of the experiment. The priming is firmly subliminal, because participants were not only unable to name objects on the first presentation, but their four-alternative forced choice (4AFC) performance was not much above chance (28.5%).

To model these phenomena, we created a response pathway with fifty states representing names of objects that are used in the experiment, e.g., chair and lamp. We also created a perceptual pathway with states representing visual patterns that correspond to the names in the response pathway. Following the experimental design, every object identity was instantiated in two distinct shapes, and every shape could be in one of nine different visual-field positions, leading to 900 distinct states in the perceptual pathway to model the possible visual stimuli. The following parameters were fit to the data. If two perceptual states, $x_i$ and $x_k$ are the same shape in different positions, they are assigned a similarity coefficient $\gamma_{ik} = 0.95$; all other similarity coefficients are zero. The association frequency, $\alpha$, for valid associations in the perceptual pathway was 22, and the response pathway 18. Other parameters were $\beta^p = .05$, $\beta^r = .01$, and $\eta = 1.0$.

The PIT model achieves a good fit to the human experimental data (Figure 2). Specifically, priming is greatest for the same shape in the same position, some priming occurs for the same shape in a different position, and no substantial priming occurs for the different shape. Figure 3a shows the time course of activation of a stimulus representation in the perceptual pathway when the stimulus is presented for 50 iterations, on both the first and third presentations. The third presentation was chosen instead of the second to make the effect of priming clearer.

Even though a shape cannot be named on the first presentation, partial information about the shape may nonetheless be available for report. The 4AFC test of Bar and Biederman provides a more sensitive measure of residual stimulus information. In past work, we modeled forced-choice tasks using a response pathway with only the alternatives under consideration. However, in this experiment, forced-choice performance must be estimated *conditional on* incorrect naming. In PIT framework, we achieve this using naming and

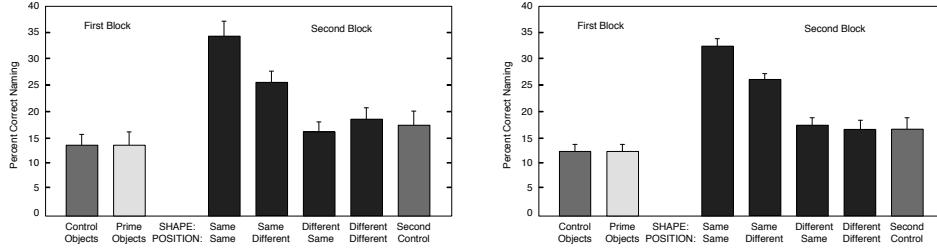

Figure 2: (left panel) Data from Bar and Biederman (1998) (right panel) Simulation of PIT. White bar: accuracy on first presentation of a prime object. Black bars: the accuracy when the object is repeated, either with the same or different shape, and in the same or different position. Grey bars: accuracy for control objects at the beginning and the end of the experiment.

forced-choice output pathways having output distributions $N(t)$ and $F(t)$, which are linked via the perceptual state, $Y^p(t)$. $F(t)$ must be reestimated with the evidence that $N(t)$ is not the target state. This inference problem is intractable. We therefore used a shortcut in which a single response pathway is used, augmented with a simple three-node belief net (Figure 3b) to capture the dependence between naming and forced choice. The belief net has a response pathway node $Y^r(t)$ connected to $F(t)$ and $N(t)$, with conditional distribution $P(N(t) = n_i | Y^r(t) = y_j) = \theta \delta_{ij} + (1 - \theta)/|Y^r|$, and an analogous distribution for $P(F(t) = f_i | Y^r(t) = y_j)$. The free parameter $\theta$ determines how veridically naming and forced-choice actions reflect response-pathway output. Over a range of $\theta$, $\theta < 1$, the model obtains forced-choice performance near chance on the first presentation when the naming response is incorrect. For example, with $\theta = 0.72$, the model produces a forced-choice accuracy on presentation 1 of 26.1%. (Interestingly, the model also produces below chance performance on presentation 2 if the object is not named correctly—23.5%—which is also found in the human data—20.0%.) Thus, by the stringent criterion of 4AFC, the model shows no access consciousness, and therefore illustrates a dissociation between priming and access consciousness. In our simulation, we followed the procedure of Bar and Biederman by including distractor alternatives with visual and semantic similarity to the target. These distractors are critical: with unrelated distractors, the model's 4AFC performance is significantly above chance, illustrating that a perceptual representation can be adequate to support some responses but not others, as Farah et al. (1994) also argued.

## 4.2 Simulation of Abrams and Greenwald (2000)

During an initial phase of the experiment, participants categorized 24 clearly visible target words as pleasant (e.g., HUMOR) or unpleasant (e.g., SMUT). They became quite familiar with the task by categorizing each word a total of eight times. In a second phase, participants were asked to classify the same targets and were given a response deadline to induce errors. The targets were preceded by masked primes that could not be identified. Of interest is the *effective valence* (or *EV*) of the target for different prime types, defined as the error rate difference between unpleasant and pleasant targets. A positive (negative) EV indicates that responses are biased toward a pleasant (unpleasant) interpretation by the prime. As one would expect, pleasant primes resulted in a positive EV, unpleasant primes in a negative EV. Of critical interest is the finding that a nonword prime formed by recombining two pleasant targets (e.g., HULIP from HUMOR and TULIP) or unpleasant targets (e.g., BIUT from BILE and SMUT) also served to bias the targets. More surprising, a positive EV resulted from unpleasant prime words formed by recombining two pleasant targets (TUMOR from TULIP and HUMOR), indicating that subliminal priming arises from word fragments, not words as unitary entities, and providing further evidence for an early locus of subliminal priming. Note that the results depend critically on the first phase of the experiment, which gave participants extensive practice on a relatively small set of words that were then used as and recombined to form primes. Words not studied in the first phase (*orphans*) provided

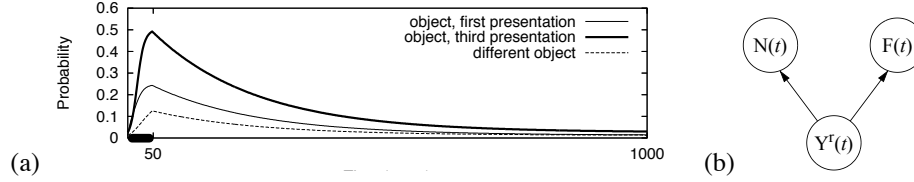

(a)                                                                              (b)

Figure 3: (a) Activation of the perceptual representation in PIT as a function of processing iterations on the first (thin solid line) and third (thick solid line) presentations of target. (b) Bayes net for performing 4AFC conditional on incorrect naming response.

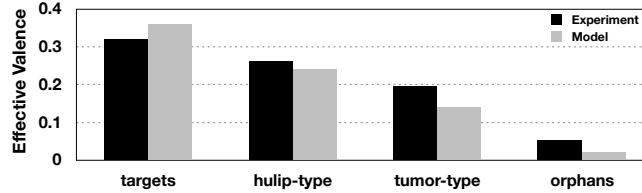

Figure 4: Effective valence of primes in the Abrams and Greenwald (2000) experiment for human subjects (black bars) and PIT model (grey bars). HULIP-type primes are almost as strong as target repetitions, and TUMOR-type primes have a positive valence, contrary to the meaning of the word.

no significant EV effect when used as primes.

In this simulation, we used a three pathway model: a perceptual pathway that maps visual patterns to orthography with 200 input states corresponding both to words, nonwords, and nonword recombinations of words; a semantic pathway that maps to 100 distinct lexical/semantic states; and a judgement pathway that maps to two responses, pleasant and unpleasant. In the perceptual pathway, similarity structure was based on letter overlap, so that HULIP was similar to both TULIP and HUMOR, with $\gamma = 0.837$. No similarity was assumed in the semantic state representation; consistent with the previous simulation, $\beta^p = .05, \beta^s = .01, \beta^j = .01$, and $\eta = .01$. At the outset of the simulation, $\alpha$ frequencies for correct associations were 15, 19, and 25 in the perceptual, semantic, and judgement pathways. The initial phase of the experiment was simulated by repeated supraliminal presentation of words, which increased the association frequencies in all three pathways through the $\Delta\alpha_{ij}$ learning rule.

Long-term supraliminal priming is essential in establishing the association strengths, as we'll explain. Short-term subliminal priming also plays a key role in the experiment. During the second phase of the experiment, residual activity from the prime—primarily in the judgement pathway—biases the response to the target. Residual activation of the prime is present even if the representation of the prime does not reach sufficient strength that it could be named or otherwise reported.

The outcome of the simulation is consistent with the human data (Figure 4). When a HULIP-type prime is presented, HUMOR and TULIP become active in the semantic pathway because of their visual similarity to HULIP. Partial activation of these two practiced words pushes the judgement pathway toward a pleasant response, resulting in a positive EV. When a TUMOR-type prime is presented, three different words become active in the semantic pathway: HUMOR, TULIP, and TUMOR itself. Although TUMOR is more active, it was not one of the words studied during the initial phase of the experiment, and as a result, it has a relatively weak association to the unpleasant judgement, in contrast to the other two words which have strong associations to the pleasant judgement. Orphan primes have little effect because they were not studied during the initial phase of the experiment, and consequently their association to pleasant and unpleasant judgements is also weak. In summary, activation of the prime along a critical, well-practiced pathway may not be sufficient to support an overt naming response, yet it may be sufficient to bias the processing of the

immediately following target.

## 5 Discussion

An important contribution of this work has been to demonstrate that specific experimental results relating to access consciousness and subliminal priming can be interpreted in a concrete computational framework. By necessity, the PIT framework, which we previously used to model supraliminal priming data, predicts the existence of subliminal priming, because the mechanisms giving rise to priming depend on degree of activation of a representation, whereas the processes giving rise to access consciousness also depend on the temporal persistence of a representation.

Another contribution of this work has been to argue that two previous computational models each tell only part of the story. Farah et al. argue that quality of representation is critical; Dehaene et al. argue that pathways to communicate representations is critical. The PIT framework argues that *both* of these features are necessary for access consciousness.

Although the PIT framework is not completely developed, it nonetheless makes a clear prediction: that subliminal priming is can never be stronger than supraliminal priming, because the maximal activation of subliminal primes is never greater than that of supraliminal primes. One might argue that many theoretical frameworks might predict the same, but no other computational model is sufficiently well developed—in terms of addressing both priming and access consciousness—to make this prediction.

In its current stage of development, a weakness of the PIT framework is that it is silent as to how perceptual and response pathways become flexibly interconnected based on task demands. However, the PIT framework is not alone in failing to address this critical issue: The Dehaene et al. model suggests that once a representation enters the global workspace, all response modules can access it, but the model does not specify how the appropriate perceptual module wins the competition to enter the global workspace, or how the appropriate response module is activated. Clearly, flexible cognitive control structures that perform these functions are intricately related to mechanisms of consciousness.

### Acknowledgments

This research was supported by NIH/IFOPAL R01 MH61549–01A1.

### References

Abrams, R. L., & Greenwald, A. G. (2000). Parts outweigh the whole (word) in unconscious analysis of meaning. *Psychological Science*, *11*(2), 118–124.

Baars, B. (1989). *A cognitive theory of consciousness.* Cambridge: Cambridge University Press.

Bar, M., & Biederman, I. (1998). Subliminal visual priming. *Psychological Science*, *9*(6), 464–468.

Block, N. (1995). On a confusion about a function of consciousness. *Brain and Behavioral Sciences*, *18*(2), 227–247.

Dehaene, S., & Naccache, L. (2001). Towards a cognitive neuroscience of consciousness: basic evidence and a workspace framework. *Cognition*, *79*, 1–37.

Dehaene, S., Sergent, C., & Changeux, J.-P. (2003). A neuronal network model linking subjective reports and objective physiological data during conscious perception. *Proceedings of the National Academy of Sciences*, *100*, 8520–8525.

Farah, M. J., O'Reilly, R. C., & Vecera, S. P. (1994). Dissociated overt and covert recognition as an emergent property of a lesioned neural network. *Psychological Review*, *100*, 571–588.

Mathis, D. W., & Mozer, M. C. (1996). Conscious and unconscious perception: a computational theory. In G. Cottrell (Ed.), *Proceedings of the Eighteenth Annual Conference of the Cognitive Science Society* (pp. 324–328). Hillsdale, NJ: Erlbaum & Associates.

Mozer, M. C., Colagrosso, M. D., & Huber, D. E. (2002). A rational analysis of cognitive control in a speeded discrimination task. In T. G. Dietterich, S. Becker, & Z. Ghahramani (Eds.), *Advances in Neural Information Processing Systems 14.* Cambridge, MA: MIT Press.

Mozer, M. C., Colagrosso, M. D., & Huber, D. E. (2003). Mechanisms of long-term repetition priming and skill refinement: A probabilistic pathway model. In *Proceedings of the Twenty-Fifth Annual Conference of the Cognitive Science Society.* Hillsdale, NJ: Erlbaum Associates.
